# The Power of Adaptivity in Identifying Statistical Alternatives

**Kevin Jamieson, Daniel Haas, Ben Recht**
University of California, Berkeley
Berkeley, CA 94720
{kjamieson,dhaas,brecht}@eecs.berkeley.edu

## Abstract

This paper studies the trade-off between two different kinds of pure exploration: breadth versus depth. We focus on the most biased coin problem, asking how many total coin flips are required to identify a "heavy" coin from an infinite bag containing both "heavy" coins with mean $\theta_1 \in (0,1)$, and "light" coins with mean $\theta_0 \in (0, \theta_1)$, where heavy coins are drawn from the bag with proportion $\alpha \in (0, 1/2)$. When $\alpha, \theta_0, \theta_1$ are unknown, the key difficulty of this problem lies in distinguishing whether the two kinds of coins have very similar means, or whether heavy coins are just extremely rare. While existing solutions to this problem require some prior knowledge of the parameters $\theta_0, \theta_1, \alpha$, we propose an adaptive algorithm that requires no such knowledge yet still obtains near-optimal sample complexity guarantees. In contrast, we provide a lower bound showing that non-adaptive strategies require at least quadratically more samples. In characterizing this gap between adaptive and nonadaptive strategies, we make connections to anomaly detection and prove lower bounds on the sample complexity of differentiating between a single parametric distribution and a mixture of two such distributions.

## 1   Introduction

The trade-off between exploration and exploitation has been an ever-present trope in the online learning literature. In contrast, this paper studies the trade-off between two different kinds of pure exploration: breadth versus depth. Consider a bag that contains an infinite number of two kinds of biased coins: "heavy" coins with mean $\theta_1 \in (0,1)$ and "light" coins with mean $\theta_0 \in (0, \theta_1)$. When a player picks a coin from the bag, with probability $\alpha$ the coin is "heavy" and with probability $(1 - \alpha)$ the coin is "light." The player can flip any coin she picks from the bag as many times as she wants, and the goal is to identify a heavy coin using as few total flips as possible. When $\alpha, \theta_0, \theta_1$ are unknown, the key difficulty of this problem lies in distinguishing whether the two kinds of coins have very similar means, or whether heavy coins are just extremely rare. That is, how does one balance flipping an individual coin many times to better estimate its mean against considering many new coins to maximize the probability of observing a heavy one. Previous work has only proposed solutions that rely on some or full knowledge $\alpha, \theta_0, \theta_1$, limiting their applicability. In this work we propose the first algorithm that requires no knowledge of $\alpha, \theta_0, \theta_1$, is guaranteed to return a heavy coin with probability at least $1 - \delta$, and flips a total number of coins, in expectation, that nearly matches known lower bounds. Moreover, our fully adaptive algorithm supports more general sub-Gaussian sources in addition to just coins, and only ever has one "coin" outside the bag at a given time, a constraint of practical importance to some applications.

In addition, we connect the most biased coin problem to anomaly detection and prove novel lower bounds on the difficulty of detecting the presence of a mixture versus just a single component of a known family of distributions (e.g. $X \sim (1-\alpha)g_{\theta_0} + \alpha g_{\theta_1}$ versus $X \sim g_\theta$ for some $\theta$). We show that in detecting the presence of a mixture distribution, there is a stark difference of difficulty

between when the underlying distribution parameters are known (e.g. $\alpha, \theta_0, \theta_1$) and when they are not. The most biased coin problem can be viewed as an online, adaptive mixture detection problem where source distributions arrive one at a time that are either $g_{\theta_0}$ with probability $(1 - \alpha)$ or $g_{\theta_1}$ with probability $\alpha$ (e.g. null or anomolous) and the player adaptively chooses how many samples to take from each distribution (to increase the signal-to-noise ratio) with the goal of identifying an anomolous distribution $f_{\theta_1}$ using as few total number of samples as possible. This work draws a contrast between the power of an adaptive versus non-adaptive (e.g. taking the same number of samples each time) approaches to this problem, specifically when $\alpha, \theta_0, \theta_1$ are unknown.

## 1.1 Motivation and Related Work for the Most Biased Coin Problem

The most biased coin problem characterizes the inherent difficulty of real-world problems including anomaly and intrusion detection and discovery of vacant frequencies in the radio spectrum. Our interest in the problem stemmed from automated hiring of crowd workers: data labeling for machine learning applications is often performed by humans, and recent work in the crowdsourcing literature accelerates labeling by organizing workers into pools of labelers and paying them to wait for incoming data [4, 12]. Workers hired on marketplaces such as Amazon's Mechanical Turk [16] vary widely in skill, and identifying high-quality workers as quickly as possible is an important challenge. We can model each worker's performance (e.g. accuracy or speed) as a random variable so that selecting a good worker is equivalent to identifying a worker with a high mean. Since we do not observe a worker's expected performance directly, we must give them tasks from which we estimate it (like repeatedly flipping a biased coin). Arlotto et al. [3] proposed a strategy with some guarantees for a related problem but did not characterize the sample complexity of the problem, the focus of our work.

The most biased coin problem was first proposed by Chandrasekaran and Karp [8]. In that work, it was shown that if $\alpha, \theta_0, \theta_1$ were known then there exists an algorithm based on the sequential probability ratio test (SPRT) that is optimal in that it minimizes the expected number of total flips to find a "heavy" coin whose posterior probability of being heavy is at least $1 - \delta$, and the expected sample complexity of this algorithm was upper-bounded by

$$\frac{16}{(\theta_1 - \theta_0)^2} \left( \frac{1 - \alpha}{\alpha} + \log \left( \frac{(1 - \alpha)(1 - \delta)}{\alpha \delta} \right) \right). \tag{1}$$

However, the practicality of the proposed algorithm is severely limited as it relies critically on knowing $\alpha, \theta_0$, and $\theta_1$ exactly. In addition, the algorithm returns to coins it has previously flipped and thus requires more than one coin to be outside the bag at a time, ruling out some applications. Malloy et al. [15] addressed some of the shortcomings of [9] (a preprint of [8]) by considering both an alternative SPRT procedure and a sequential thresholding procedure. Both of these proposed algorithms only ever have one coin out of the bag at a time. However, the former requires knowledge of all relevant parameters $\alpha, \theta_0, \theta_1$, and the latter requires knowledge of $\alpha, \theta_0$. Moreover, these results are only presented for the asymptotic case where $\delta \to 0$.

The most biased coin problem can be viewed through the lens of multi-armed bandits. In the best-arm identification problem, the player has access to $K$ distributions (arms) such that if arm $i \in [K]$ is sampled (pulled), an iid random variable with mean $\mu_i$ is observed; the objective is to identify the arm associated with the highest mean with probability at least $1 - \delta$ using as few pulls as possible (see [14] for a short survey). In the *infinite* armed bandit problem, the player is not confined to $K$ arms but an infinite reservoir of arms such that a draw from this reservoir results in an arm with a mean $\mu$ drawn from some distribution; the objective is to identify the highest mean possible after $n$ total pulls for any $n > 0$ with probability $1 - \delta$ (see [7]). The most biased coin problem is an instance of this latter game with the arm reservoir distribution of means $\mu$ defined as $\mathbb{P}(\mu \geq \theta_1 - \epsilon) = \alpha \mathbf{1}_{\epsilon \geq 0} + (1 - \alpha)\mathbf{1}_{\epsilon \geq \theta_1 - \theta_0}$ for all $\epsilon$. Previous work has focused on an alternative arm distribution reservoir that satisfies $E\epsilon^\beta \leq \mathbb{P}(\mu \geq \mu_* - \epsilon) \leq E'\epsilon^\beta$ for some $\mu_* \in [0, 1]$ where $E, E'$ are constants and $\beta$ is known [5, 21, 6, 7]. Because neither arm distribution reservoir can be written in terms of the other, neither work subsumes the other. Note that one can always apply an algorithm designed for the infinite armed bandit problem to any finite $K$-armed bandit problem by defining the arm reservoir as placing a uniform distribution over the $K$ arms. This is appealing when $K$ is very large and one wishes to guarantee nontrivial performance when the number of pulls is much less than $K^1$. The most biased problem is a special case of the $K$-armed reservoir distribution where one arm has mean $\theta_1$ and $K - 1$ arms have mean $\theta_0$ with $\alpha = \frac{1}{K}$.

Given that [8] and [15] are provably optimal algorithms for the most biased coin problem given knowledge of $\alpha, \theta_0, \theta_1$, it is natural to consider a procedure that first estimates these unknown parameters first and then uses these estimates in the algorithms of [8] or [15]. Indeed, in the $\beta$-parameterized arm reservoir setting discussed above, this is exactly what Carpentier and Valko [7] propose to do, suggesting a particular estimator for $\beta$ given a lower bound $\widehat{\beta} \leq \beta$. They show that this estimator is sufficient to obtain the same sample complexity result up to log factors as when $\beta$ was known. Sadly, through upper and lower bounds we show that for the most biased coin problem this *estimate-then-explore* approach requires quadratically more flips than our proposed algorithm that adapts to these unknown parameters. Specifically, we show that when $\theta_1 - \theta_0$ is sufficiently small one cannot use a static estimation step to determine whether $\alpha = 0$ or $\alpha > 0$ unless a number of samples *quadratic* in the optimal sample complexity are taken.

Our contributions to the most biased coin problem include a novel algorithm that never has more than one coin outside the bag at a time, has no knowledge of the distribution parameters, supports distributions on $[0, 1]$ rather than just "coins," and comes within log factors of the known information-theoretic lower bound and Equation 1 which is achieved by an algorithm that knows the parameters. See Table 1 for an overview of the upper and lower bounds proved in this work for this problem. We believe that our algorithm is the first solution to the most biased coin problem that does not require prior knowledge of the problem parameters and that the same approach can be reworked to solve more general instances of the infinite-armed bandit problem, including the $\beta$-parameterized and $K$-armed reservoir cases described of above. Finally, if an algorithm is desired for arbitrary arm reservoir distributions, this work rules out an estimate-then-explore approach.

## 1.2 Problem Statement

Let $\theta \in \Theta$ index a family of single-parameter probability density functions $g_\theta$ and fix $\theta_0, \theta_1 \in \Theta$, $\alpha \in [0, 1/2]$. For any $\theta \in \Theta$ assume that $g_\theta$ is known to the procedure. Note that in the most biased coin problem, $g_\theta =$Bernoulli$(\theta)$, but in general it is arbitrary (e.g. $\mathcal{N}(\theta, 1)$). Consider a sequence of iid Bernoulli random variables $\xi_i \in \{0, 1\}$ for $i = 1, 2, \ldots$ where each $\mathbb{P}(\xi_i = 1) = 1 - \mathbb{P}(\xi_i = 0) = \alpha$. Let $X_{i,j}$ for $j = 1, 2, \ldots$ be a sequence of random variables drawn from $g_{\theta_1}$ if $\xi_i = 1$ and $g_{\theta_0}$ otherwise, and let $\{\{X_{i,j}\}_{j=1}^{M_i}\}_{i=1}^N$ represent the sampling history generated by a procedure for some $N \in \mathbb{N}$ and $(M_1, \ldots, M_N) \in \mathbb{N}^N$. Any valid procedure behaves accordingly:

---

**Algorithm 1** The most biased coin problem definition. Only the last distribution drawn may be sampled or declared heavy, enforcing the rule that only one coin may be outside the bag at a time.

---

**Initialize** an empty history ($N = 1, M = (0, 0, \ldots)$).
**Repeat** until heavy distribution declared:
    **Choose** one of
        1. draw a sample from distribution $N$, $M_N \leftarrow M_N + 1$
        2. draw a sample from the $(N + 1)$st distribution, $M_{N+1} = 1$, $N \leftarrow N + 1$
        3. declare distribution $N$ as heavy

---

**Definition 1** *We say a strategy for the most biased coin problem is $\delta$-**probably correct** if for all $(\alpha, \theta_0, \theta_1)$ it identifies a "heavy" $g_{\theta_1}$ distribution with probability at least $1 - \delta$.*

**Definition 2 (Strategies for the most biased coin problem)** *An **estimate-then-explore** strategy is a strategy that, for any fixed $m \in \mathbb{N}$, begins by sampling each successive coin exactly $m$ times for a number of coins that is at least the minimum necessary for any test to determine that $\alpha \neq 0$ with probability at least $1 - \delta$, then optionally continues sampling with an arbitrary strategy that declares a heavy coin. An **adaptive strategy** is any strategy that is not an estimate-then-explore strategy.*

We study the *estimate-then-explore strategy* because there exist optimal algorithms [8, 15] for the most biased coin problem if $\alpha, \theta_0, \theta_1$ are known, so it is natural to consider estimating these quantities then using one of these algorithms. Note that the algorithm of [7] for the $\beta$-parameterized infinite armed bandit problem discussed above can be considered an *estimate-then-explore strategy* since it first estimates $\beta$ by sampling a fixed number of samples from a set of arms, and then uses this estimate to draw a fixed number of arms and applies a UCB-style algorithm to these arms. A contribution of this work is showing that such a strategy is infeasible for the most biased coin problem.

For all strategies that are $\delta$-probably correct and follow the interface of Algorithm 1, our goal is to provide lower and upper bounds on the quantity $\mathbb{E}[T] := \mathbb{E}[\sum_{i=1}^{N} M_i]$ for any $(\alpha, \theta_0, \theta_1)$ if $N$ denotes the final number of coins considered.

## 2 From Identifying Coins to Detecting Mixture Distributions

Addressing the most biased coin problem, [15] analyzes perhaps the most natural strategy: fix an $m \in \mathbb{N}$ and flip each successive coin exactly $m$ times. The relevant questions are how large does $m$ have to be in order to guarantee correctness with probability $1 - \delta$, and for a given $m$ how long must one wait to declare a "heavy" coin? The authors partially answer these questions and we improve upon them (see Section 3.2.1) which leads us to our study of the difficulty of detecting the presence of a mixture distribution. As an example of the kind of lower bounds shown in this work, if we observe a sequence of random variables $X_1, \ldots, X_n$, consider the following hypothesis test:

$$\begin{aligned} \mathbf{H}_0 &: \forall i \ X_1, \ldots, X_n \sim \mathcal{N}(\theta, \sigma^2) \quad \text{for some } \theta \in \mathbb{R}, \\ \mathbf{H}_1 &: \forall i \ X_1, \ldots, X_n \sim (1 - \alpha)\mathcal{N}(\theta_0, \sigma^2) + \alpha \, \mathcal{N}(\theta_1, \sigma^2) \end{aligned} \tag{P1}$$

which will henceforth be referred to as Problem P1 or just (P1). We can show that if $\theta_0, \theta_1, \alpha$ are *known* and $\theta = \theta_0$, then it is *sufficient* to observe just $\max\{1/\alpha, \frac{\sigma^2}{\alpha^2(\theta_1-\theta_0)^2} \log(1/\delta)\}$ samples to determine the correct hypothesis with probability at least $1 - \delta$. However, if $\theta_0, \theta_1, \alpha$ are *unknown* then it is *necessary* to observe at least $\max\left\{1/\alpha, \left(\frac{\sigma^2}{\alpha(\theta_1-\theta_0)^2}\right)^2 \log(1/\delta)\right\}$ samples in expectation whenever $\frac{(\theta_1-\theta_0)^2}{\sigma^2} \leq 1$ and $\max\{1/\alpha, \frac{\sigma^2}{\alpha^2(\theta_1-\theta_0)^2} \log(1/\delta)\}$ otherwise (see Appendix C).

Recognizing $\frac{(\theta_1-\theta_0)^2}{\sigma^2}$ as the KL divergence between two Gaussians of $\mathbf{H}_1$, we observe startling consequences for anomaly detection when the parameters of the underlying distributions are unknown: if the anomalous distribution is well separated from the null distribution, then detecting an anomalous component is only about as hard as observing just one anomalous sample (i.e. $1/\alpha$) multiplied by the inverse KL divergence between the null and anomalous distributions. However, when the two distributions are *not* well separated then the necessary sample complexity explodes to this latter quantity *squared*. In Section 4 we will investigate adaptive methods for dramatically decreasing this sample complexity.

Our lower bounds are based on the detection of the presence of a mixture of two distributions of an exponential family versus just a single distribution of the same family. There has been extensive work in the estimation of mixture distributions [13, 11] but this literature often assumes that the mixture coefficient $\alpha$ is bounded away from 0 and 1 to ensure a sufficient number of samples from each distribution. In contrast, we highlight the regime when $\alpha$ is arbitrarily small, as is the case in statistical anomaly detection [10, 20, 2]. Property testing, e.g. unimodality, [1] is relevant but can lack interpetability or strength in favor of generality. Considering the exponential family allowing us to make interpretable statements about the relevant problem parameters in different regimes.

**Preliminaries** Let $P$ and $Q$ be two probability distributions with densities $p$ and $q$, respectively. For simplicity, assume $p$ and $q$ have the same support. Define the *KL Divergence* between $P$ and $Q$ as $KL(P, Q) = \int \log\left(\frac{p(x)}{q(x)}\right) dp(x)$. Define the $\chi^2$ *Divergence* between $P$ and $Q$ as $\chi^2(P, Q) = \int \left(\frac{p(x)}{q(x)} - 1\right)^2 dq(x) = \int \frac{(p(x)-q(x))^2}{q(x)} dx$. Note that by Jensen's inequality

$$KL(P, Q) = \mathbb{E}_p\left[\log\left(\frac{p}{q}\right)\right] \leq \log\left(\mathbb{E}_p\left[\frac{p}{q}\right]\right) = \log\left(\chi^2(P, Q) + 1\right) \leq \chi^2(P, Q). \tag{2}$$

Examples: If $P = \mathcal{N}(\theta_1, \sigma^2)$ and $Q = \mathcal{N}(\theta_0, \sigma^2)$ then $KL(P, Q) = \frac{(\theta_1-\theta_0)^2}{2\sigma^2}$ and $\chi^2(P, Q) = e^{\frac{(\theta_1-\theta_0)^2}{\sigma^2}} - 1$. If $P = \text{Bernoulli}(\theta_1)$ and $Q = \text{Bernoulli}(\theta_0)$ then $KL(P, Q) = \theta_1 \log(\frac{\theta_1}{\theta_0}) + (1 - \theta_1) \log(\frac{1-\theta_1}{1-\theta_0}) \leq \frac{(\theta_1-\theta_0)^2/2}{\theta_0(1-\theta_0) - [(\theta_1-\theta_0)(2\theta_0-1)]_+}$ and $\chi^2(P, Q) = \frac{(\theta_1-\theta_0)^2}{\theta_0(1-\theta_0)}$. All proofs appear in the appendix.

## 3 Lower bounds

We present lower bounds on the sample complexity of $\delta$-*probably correct* strategies for the most biased coin problem that follow the interface of Algorithm 1. Lower bounds are stated for any

*adaptive* strategy in Section 3.1, non-adaptive strategies that may have knowledge of the parameters but sample each distribution the same number of times in Section 3.2.1, and *estimate-then-explore* strategies that do not have prior knowledge of the parameters in Section 3.2.2. Our lower bounds, with the exception of the adaptive strategy, are based on the difficulty of detecting the presence of a mixture distribution, and this reduction is explained in Section 3.2.

## 3.1 Adaptive strategies

The following theorem, reproduced from [15], describes the sample complexity of any $\delta$-probably correct algorithm for the most biased coin identification problem. Note that this lower bound holds for any procedure even if it returns to previously seen distributions to draw additional samples and even if it knows $\alpha, \theta_0, \theta_1$.

**Theorem 1** *[15, Theorem 2] Fix $\delta \in (0,1)$. Let $T$ be the total number of samples taken of any procedure that is $\delta$-probably correct in identifying a heavy distribution. Then*

$$\mathbb{E}[T] \geq c_1 \max \left\{ \frac{1-\delta}{\alpha}, \frac{(1-\delta)}{\alpha KL(g_{\theta_0}|g_{\theta_1})} \right\}$$

*whenever $\alpha \leq c_2 \delta$ where $c_1, c_2 \in (0,1)$ are absolute constants.*

The above theorem is directly applicable to the special case where $g_\theta$ is a Bernoulli distribution, implying a lower bound of $\max \left\{ \frac{1-\delta}{\alpha}, \frac{2 \min\{\theta_0(1-\theta_0), \theta_1(1-\theta_1)\}}{\alpha(\theta_1-\theta_0)^2} \right\}$ for the most biased coin problem. The upper bounds of our proposed procedures for the most biased coin problem presented later will be compared to this benchmark.

## 3.2 The detection of a mixture distribution and the most biased coin problem

First observe that identifying a specific distribution $i \leq N$ as heavy (i.e. $\xi_i = 1$) or determining that $\alpha$ is strictly greater than 0, is at least as hard as detecting that *any* of the distributions up to distribution $N$ is heavy. Thus, a lower bound on the total expected number of samples of all considered distributions for this strictly easier detection problem is also a lower bound for the estimate-then-explore strategy for the most biased coin identification problem.

The estimate-then-explore strategy fixes an $m \in \mathbb{N}$ prior to starting the game and then samples each distribution exactly $m$ times, i.e. $M_i = m$ for all $i \leq N$ for some $N$. To simplify notation let $f_\theta$ denote the distribution of the sufficient statistics of these $m$ samples. In general $f_\theta$ is a product distribution, but when $g_\theta$ is a Bernoulli distribution, as in the biased coin problem, we can take $f_\theta$ to be a Binomial distribution with parameters $(m, \theta)$. Now our problem is more succinctly described as:

$$\mathbf{H}_0 : \forall i \ X_i \sim f_\theta \quad \text{for some } \theta \in \widetilde{\Theta} \subseteq \Theta,$$

$$\mathbf{H}_1 : \forall i \ \xi_i \sim \text{Bernoulli}(\alpha), \quad \forall i \ X_i \sim \begin{cases} f_{\theta_0} & \text{if } \xi_i = 0 \\ f_{\theta_1} & \text{if } \xi_i = 1 \end{cases} \tag{P2}$$

If $\theta_0$ and $\theta_1$ are close to each other, or if $\alpha$ is very small, it can be very difficult to decide between $\mathbf{H}_0$ and $\mathbf{H}_1$ even if $\alpha, \theta_0, \theta_1$ are known a priori. Note that when the parameters are *known*, one can take $\widetilde{\Theta} = \{\theta_0\}$. However, when the parameters are *unknown*, one takes $\widetilde{\Theta} = \Theta$ to prove a lower bound on the sample complexity of the estimate-then-explore algorithm, which is tasked with deciding whether or not samples are coming from a mixture of distributions or just a single distribution within the family. That is, lower bounds on the sample complexity when the parameters are known and unknown follow by analyzing a simple binary and composite hypothesis test, respectively. In what follows, for any event $A$, let $\mathbb{P}_i(A)$ and $\mathbb{E}_i[A]$ denote probability and expectation of $A$ under hypothesis $\mathbf{H}_i$ for $i \in \{0,1\}$ (the specific value of $\theta$ in $\mathbf{H}_0$ will be clear from context). The next claim is instrumental in our ability to prove lower bounds on the difficulty of the hypothesis tests.

**Claim 1** *Any procedure that is $\delta$-probably correct also satisfies $\mathbb{P}_0(N < \infty) \leq \delta$ whenever $\alpha = 0$.*

### 3.2.1 Sample complexity when parameters are known

**Theorem 2** *Fix $\delta \in (0,1)$. Consider the hypothesis test of Problem P2 for any fixed $\theta \in \widetilde{\Theta} \subseteq \Theta$. Let $N$ be the random number of distributions considered before stopping and declaring a*

*hypothesis. If a procedure satisfies $\mathbb{P}_0(N < \infty) \leq \delta$ and $\mathbb{P}_1(\cup_{i=1}^N \{\xi_i = 1\}) \geq 1 - \delta$, then $\mathbb{E}_1[N] \geq \max\left\{\frac{1-\delta}{\alpha}, \frac{\log(1/\delta)}{KL(\mathbb{P}_1|\mathbb{P}_0)}\right\} \geq \max\left\{\frac{1-\delta}{\alpha}, \frac{\log(1/\delta)}{\chi^2(\mathbb{P}_1|\mathbb{P}_0)}\right\}$. In particular, if $\widetilde{\Theta} = \{\theta_0\}$ then*

$$\mathbb{E}_1[N] \geq \max\left\{\frac{1-\delta}{\alpha}, \frac{\log(1/\delta)}{\alpha^2 \chi^2(f_{\theta_1}|f_{\theta_0})}\right\}.$$

The next corollary relates Theorem 2 to the most biased coin problem and is related to Malloy et al. [15, Theorem 4] that considers the limit as $\alpha \to 0$ and assumes $m$ is sufficiently large (specifically, large enough for the Chernoff-Stein lemma to apply). In contrast, our result holds for all finite $\delta, \alpha, m$.

**Corollary 1** *Fix $\delta \in (0,1)$. For any $m \in \mathbb{N}$ consider a $\delta$-probably correct strategy that flips each coin exactly $m$ times. If $N_m$ is the number of coins considered before declaring a coin as heavy then*

$$\min_{m \in \mathbb{N}} \mathbb{E}[mN_m] \geq \frac{(1-\delta)\log\left(\frac{\log(1/\delta)}{\alpha}\right)}{\alpha} \frac{\theta_0(1-\theta_0)}{(\theta_1 - \theta_0)^2}.$$

One can show the existence of such a strategy with a nearly matching upperbound when $\alpha, \theta_0, \theta_1$ are known (see Appendix B.1). Note that this is at least $\log(1/\alpha)$ larger than the sample complexity of (1) that can be achieved by an adaptive algorithm when the parameters are known.

### 3.2.2 Sample complexity when parameters are unknown

If $\alpha$, $\theta_0$, and $\theta_1$ are unknown, we cannot test $f_{\theta_0}$ against the mixture $(1-\alpha)f_{\theta_0} + \alpha f_{\theta_1}$. Instead, we have the general composite test of *any* individual distribution against *any* mixture, which is at least as hard as the hypothesis test of Problem P2 with $\widetilde{\Theta} = \{\theta\}$ for some particular worst-case setting of $\theta$. Without any specific form of $f_\theta$, it is difficult to pick a worst case $\theta$ that will produce a tight bound. Consequently, in this section we consider single parameter exponential families (defined formally below) to provide us with a class of distributions in which we can reason about different possible values for $\theta$. Since exponential families include Bernoulli, Gaussian, exponential, and many other distributions, the following theorem is general enough to be useful in a wide variety of settings. The constant $C$ referred to in the next theorem is an absolute constant under certain conditions that we outline in the following remark and corollary, its explicit form is given in the proof.

**Theorem 3** *Suppose $f_\theta$ for $\theta \in \Theta \subset \mathbb{R}$ is a single parameter exponential family so that $f_\theta(x) = h(x)\exp(\eta(\theta)x - b(\eta(\theta)))$ for some scalar functions $h, b, \eta$ where $\eta$ is strictly increasing. If $\widetilde{\Theta} = \{\theta_*\}$ where $\theta_* = \eta^{-1}((1-\alpha)\eta(\theta_0) + \alpha\eta(\theta_1))$ and $N$ is the stopping time of any procedure that satisfies $\mathbb{P}_0(N < \infty) \leq \delta$ and $\mathbb{P}_1(\cup_{i=1}^N \{\xi_i = 1\}) \geq 1 - \delta$, then*

$$\mathbb{E}_1[N] \geq \max\left\{\frac{1-\delta}{\alpha}, \frac{\log(\frac{1}{\delta})}{C\left(\frac{1}{2}\alpha(1-\alpha)(\eta(\theta_1)-\eta(\theta_0))^2\right)^2}\right\}.$$

*where $C$ is a constant that may depend on $\alpha, \theta_0, \theta_1$.*

The following remark and corollary apply Theorem 3 to the special cases of Gaussian mixture model detection and the most biased coin problem, respectively.

**Remark 1** *When $\alpha, \theta_0, \theta_1$ are unknown, any procedure has no knowledge of $\widetilde{\Theta}$ in Problem P2 and consequently it cannot rule out $\theta = \theta_*$ for $\mathbf{H}_0$ where $\theta_*$ is defined in Theorem 3. If $f_\theta = \mathcal{N}(\theta, \sigma^2)$ for known $\sigma$, then whenever $\frac{(\theta_1-\theta_0)^2}{\sigma^2} \leq 1$ the constant $C$ in Theorem 3 is an absolute constant and consequently, $\mathbb{E}_1[N] = \Omega\left(\left(\frac{\sigma^2}{\alpha(\theta_1-\theta_0)^2}\right)^2 \log(1/\delta)\right)$. Conversely, when $\alpha, \theta_0, \theta_1$ are known, then we simply need to determine whether samples came from $\mathcal{N}(\theta_0, \sigma^2)$ or $(1-\alpha)\mathcal{N}(\theta_0, \sigma^2) + \alpha\mathcal{N}(\theta_1, \sigma^2)$, and we show that it is sufficient to take just $O\left(\frac{\sigma^2}{\alpha^2(\theta_1-\theta_0)^2}\log(1/\delta)\right)$ samples (see Appendix C).*

**Corollary 2** *Fix $\delta \in [0,1]$ and assume $\theta_0, \theta_1$ are bounded sufficiently far from $\{0,1\}$ such that $2(\theta_1 - \theta_0) \leq \min\{\theta_0(1-\theta_0), \theta_1(1-\theta_1)\}$. For any $m$ let $N_m$ be the number of coins a $\delta$-probably correct estimate-then-explore strategy that flips each coin $m$ times in the exploration step. Then*

$$m\mathbb{E}[N_m] \geq \frac{c'\min\{\frac{1}{m}, \theta_*(1-\theta_*)\}}{\left(\alpha(1-\alpha)\frac{(\theta_1-\theta_0)^2}{\theta_*(1-\theta_*)}\right)^2}\log(\tfrac{1}{\delta}) \quad \text{whenever} \quad m \leq \frac{\theta_*(1-\theta_*)}{(\theta_1-\theta_0)^2}.$$

*where $c'$ is an absolute constant and $\theta_* = \eta^{-1}((1-\alpha)\eta(\theta_0) + \alpha\eta(\theta_1)) \in [\theta_0, \theta_1]$.*

**Remark 2** *If $\alpha, \theta_0, \theta_1$ are unknown, any estimate-then-explore strategy (or the strategy described in Corollary 1) would be unable to choose an $m$ that depended on these parameters, so we can treat it as a constant. Thus, for the case when $\theta_0$ and $\theta_1$ are bounded away from $\{0, 1\}$ (e.g. $\theta_0, \theta_1 \in [1/8, 7/8]$), the above corollary states that for any fixed $m$, whenever $\theta_1 - \theta_0$ is sufficiently small the number of samples necessary for these strategies to identify a heavy coin scales like $\left(\frac{1}{\alpha(\theta_1 - \theta_0)^2}\right)^2 \log(1/\delta)$. This is striking example of the difference when parameters are known versus when they are not and effectively rules out an estimate-then-explore strategy for practical purposes.*

| Setting | Upper Bound | | Lower Bound | |
|---|---|---|---|---|
| Fixed, known $\alpha, \theta_0, \theta_1$ | $\frac{\log(1/(\delta\alpha))}{\alpha\epsilon^2}$, | Thm. 7 | $\frac{\log(\log(1/\delta)/\alpha)}{\alpha\epsilon^2}$ | Cor. 1 |
| Adaptive, known $\alpha, \theta_0, \theta_1$ | $\frac{1}{\epsilon^2}\left(\frac{1}{\alpha} + \log(\frac{1}{\delta})\right)$ | [8, 15], Thm. 4 | $\frac{1}{\alpha\epsilon^2}$ | [15] |
| Est+Expl, unknown $\alpha, \theta_0, \theta_1$ | Unconsidered$^\dagger$ | | $\left(\frac{1}{\alpha\epsilon^2}\right)^2 \log(\frac{1}{\delta})$ | Cor. 2 |
| Adaptive, unknown $\alpha, \theta_0, \theta_1$ | $\frac{c \log\left(\frac{1}{\alpha\epsilon^2}\right) \log\left(\log\left(\frac{1}{\alpha\epsilon^2}\right)/\delta\right)}{\alpha\epsilon^2}$ | Thm. 5 | $\frac{1}{\alpha\epsilon^2}$ | [15] |

Table 1: Upper and lower bounds on the expected sample complexity of different $\delta$-probably correct strategies. Fixed refers to the strategy of Corollary 1. For this table, we assume $\min\{\theta_0(1 - \theta_0), \theta_1(1 - \theta_1)\}$ is lower bounded by a constant (e.g. $\theta_0, \theta_1 \in [1/8, 7/8]$) and $\epsilon = \theta_1 - \theta_0$ is sufficiently small. Also note that the upperbounds apply to distributions supported on $[0, 1]$, not just coins. All results without bracketed citations were unknown prior to this work. † Due to our discouraging lower bound for any estimate-then-explore strategy, it is inadvisable to propose an algorithm.

## 4 Near optimal adaptive algorithm

In this section we propose an algorithm that has no prior knowledge of the parameters $\alpha, \theta_0, \theta_1$ yet yields an upper bound that matches the lower bound of Theorem 1 up to logarithmic factors. We assume that samples from heavy or light distributions are supported on $[0, 1]$, and that drawn samples are independent and unbiased estimators of the mean, i.e., $\mathbb{E}[X_{i,j}] = \mu_i$ for $\mu_i \in \{\theta_0, \theta_1\}$. All results can be easily extended to sub-Gaussian distributions. Consider Algorithm 2, an SPRT-like procedure [18] for finding a heavy distribution given $\delta$ and lower bounds on $\alpha$ and $\epsilon = \theta_1 - \theta_0$. It improves upon prior work by supporting arbitrary distributions on $[0, 1]$ and requires only bounds $\alpha, \epsilon$.

---

**Algorithm 2** Adaptive strategy for heavy distribution identification with inputs $\alpha_0, \epsilon_0, \delta$

---

**Given** $\delta \in (0, 1/4), \alpha_0 \in (0, 1/2), \epsilon_0 \in (0, 1)$.
**Initialize** $n = \lceil 2\log(9)/\alpha_0 \rceil, m = \lceil 64\epsilon_0^{-2} \log(14n/\delta) \rceil, A = -8\epsilon_0^{-1} \log(21)$,
$\quad B = 8\epsilon_0^{-1} \log(14n/\delta), k_1 = 5, k_2 = \lceil 8\epsilon_0^{-2} \log(2k_1/\min\{\delta/8, m^{-1}\epsilon_0^{-2}\}) \rceil$.
**Draw** $k_1$ distributions and sample them each $k_2$ times.
**Estimate** $\widehat{\theta_0} = \min_{i=1,\dots,k_1} \widehat{\mu}_{i,k_2}, \hat{\gamma} = \widehat{\theta_0} + \epsilon_0/2$.
**Repeat** for $i = 1, \dots, n$:
$\quad$**Draw** distribution $i$.
$\quad$**Repeat** for $j = 1, \dots, m$:
$\quad\quad$**Sample** distribution $i$ and observe $X_{i,j}$.
$\quad\quad$**If** $\sum_{k=1}^{j}(X_{i,k} - \hat{\gamma}) > B$:
$\quad\quad\quad$**Declare** distribution $i$ to be heavy and **Output** distribution $i$.
$\quad\quad$**Else if** $\sum_{k=1}^{j}(X_{i,k} - \hat{\gamma}) < A$:
$\quad\quad\quad$**break**.
**Output** null.

---

**Theorem 4** *If Algorithm 2 is run with $\delta \in (0, 1/4), \alpha_0 \in (0, 1/2), \epsilon_0 \in (0, 1)$, then the expected number of total samples taken by the algorithm is no more than*

$$\frac{c'\alpha\log(1/\alpha_0) + c''\log\left(\frac{1}{\delta}\right)}{\alpha_0\epsilon_0^2} \tag{3}$$

*for some absolute constants $c'$, $c''$, and all of the following hold: 1) with probability at least $1 - \delta$, a light distribution is not returned, 2) if $\epsilon_0 \leq \theta_1 - \theta_0$ and $\alpha_0 \leq \alpha$, then with probability $\frac{4}{5}$ a heavy distribution is returned, and 3) the procedure takes no more than $\frac{c \log(1/(\alpha_0 \delta))}{\alpha_0 \epsilon_0^2}$ total samples.*

The second claim of the theorem holds only with constant probability (versus with probability $1 - \delta$) since the probability of observing a heavy distribution among the $n = \lceil 2 \log(4)/\alpha_0 \rceil$ distributions only occurs with constant probability. One can show that if the outer loop of algorithm is allowed to run indefinitely (with $m$ and $n$ defined as is), $\epsilon_0 = \theta_1 - \theta_0$, $\alpha_0 = \alpha$, and $\widehat{\theta}_0 = \theta_0$, then a heavy coin is returned with probability at least $1 - \delta$ and the expected number of samples is bounded by (3). If a tight lower bound is known on either $\epsilon = \theta_1 - \theta_0$ or $\alpha$, there is only one parameter that is unknown and the "doubling trick", along with Theorem 4, can be used to identify a heavy coin with just $\frac{\log(\log(\epsilon^{-2})/\delta)}{\alpha \epsilon^2}$ and $\frac{\log(\log(\alpha^{-1})/\delta)}{\alpha \epsilon^2}$ samples, respectively (see Appendix B.3).

Now consider Algorithm 3 that assumes no prior knowledge of $\alpha, \theta_0, \theta_1$, the first result for this setting that we are aware of. We remark that while the placing of "landmarks" $(\alpha_k, \epsilon_k)$ throughout the search space as is done in Algorithm 3 appears elementary in hindsight, it is surprising that so few can cover this two dimensional space since one has to balance the exploration of $\alpha$ and $\epsilon$. We believe similar a similar approach may be generalized for more generic infinite armed bandit problems.

---

**Algorithm 3** Adaptive strategy for heavy distribution identification with unknown parameters

---

**Given** $\delta > 0$.
**Initialize** $\ell = 1$, heavy distribution $h = $ null.
**Repeat** until $h$ is not null:
    **Set** $\gamma_\ell = 2^\ell, \delta_\ell = \delta/(2\ell^3)$
    **Repeat** for $k = 0, \ldots, \ell$:
        **Set** $\alpha_k = \frac{2^k}{\gamma_\ell}, \epsilon_k = \sqrt{\frac{1}{2\alpha_k \gamma_\ell}}$
        **Run** Algorithm 2 with $\alpha_0 = \alpha_k, \epsilon_0 = \epsilon_k, \delta = \delta_\ell$ and **Set** $h$ to its output.
        **If** $h$ is not null **break**
    **Set** $\ell = \ell + 1$
**Output** $h$

---

**Theorem 5 (Unknown $\alpha, \theta_0, \theta_1$)** *Fix $\delta \in (0, 1)$. If Algorithm 3 is run with $\delta$ then with probability at least $1 - \delta$ a heavy distribution is returned and the expected number of total samples taken is bounded by*

$$c \frac{\log_2(\frac{1}{\alpha \epsilon^2})}{\alpha \epsilon^2} (\alpha \log_2(\frac{1}{\epsilon^2}) + \log(\log_2(\frac{1}{\alpha \epsilon^2})) + \log(1/\delta))$$

*for an absolute constant c.*

## 5 Conclusion

While all prior works have required at least partial knowledge of $\alpha, \theta_0, \theta_1$ to solve the most biased coin problem, our algorithm requires no knowledge of these parameters yet obtain the near-optimal sample complexity. In addition, we have proved lower bounds on the sample complexity of detecting the presence of a mixture distribution when the parameters are known or unknown, with consequences for any estimate-then-explore strategy, an approach previously proposed for an infinite armed bandit problem. Extending our adaptive algorithm to arbitrary arm reservoir distributions is of significant interest. We believe a successful algorithm in this vein could have a significant impact on how researchers think about sequential decision processes in both finite and uncountable action spaces.

**Acknowledgments** Kevin Jamieson is generously supported by ONR awards N00014-15-1-2620, and N00014-13-1-0129. This research is supported in part by NSF CISE Expeditions Award CCF-1139158, DOE Award SN10040 DE-SC0012463, and DARPA XData Award FA8750-12-2-0331, and gifts from Amazon Web Services, Google, IBM, SAP, The Thomas and Stacey Siebel Foundation, Apple Inc., Arimo, Blue Goji, Bosch, Cisco, Cray, Cloudera, Ericsson, Facebook, Fujitsu, Guavus, HP, Huawei, Intel, Microsoft, Pivotal, Samsung, Schlumberger, Splunk, State Farm and VMware.

## Footnotes

[1] All algorithms for $K$-armed bandit problem known to these authors begins by sampling each arm once so that until the number of pulls exceeds $K$, performance is no better than random selection.

# References

[1] Jayadev Acharya, Constantinos Daskalakis, and Gautam C Kamath. Optimal testing for properties of distributions. In *Advances in Neural Information Processing Systems*, pages 3577–3598, 2015.

[2] Deepak Agarwal. Detecting anomalies in cross-classified streams: a bayesian approach. *Knowledge and Information Systems*, 11(1):29–44, 2006.

[3] Alessandro Arlotto, Stephen E Chick, and Noah Gans. Optimal hiring and retention policies for heterogeneous workers who learn. *Management Science*, 60(1):110–129, 2013.

[4] Michael S Bernstein, Joel Brandt, Robert C Miller, and David R Karger. Crowds in two seconds: enabling realtime crowd-powered interfaces. *UIST*, 2011.

[5] Donald A. Berry, Robert W. Chen, Alan Zame, David C. Heath, and Larry A. Shepp. Bandit problems with infinitely many arms. *Ann. Statist.*, 25(5):2103–2116, 10 1997.

[6] Thomas Bonald and Alexandre Proutiere. Two-target algorithms for infinite-armed bandits with bernoulli rewards. In C.J.C. Burges, L. Bottou, M. Welling, Z. Ghahramani, and K.Q. Weinberger, editors, *Advances in Neural Information Processing Systems 26*, pages 2184–2192. Curran Associates, Inc., 2013.

[7] Alexandra Carpentier and Michal Valko. Simple regret for infinitely many armed bandits. *arXiv preprint arXiv:1505.04627*, 2015.

[8] Karthekeyan Chandrasekaran and Richard Karp. Finding a most biased coin with fewest flips. In *Proceedings of The 27th Conference on Learning Theory*, pages 394–407, 2014.

[9] Karthekeyan Chandrasekaran and Richard M. Karp. Finding the most biased coin with fewest flips. *CoRR*, abs/1202.3639, 2012. URL http://arxiv.org/abs/1202.3639.

[10] Eleazar Eskin. Anomaly detection over noisy data using learned probability distributions. In *Proceedings of the Seventeenth International Conference on Machine Learning*, ICML '00, pages 255–262, San Francisco, CA, USA, 2000. Morgan Kaufmann Publishers Inc.

[11] Yoav Freund and Yishay Mansour. Estimating a mixture of two product distributions. In *Proceedings of the twelfth annual conference on Computational learning theory*, pages 53–62. ACM, 1999.

[12] Daniel Haas, Jiannan Wang, Eugene Wu, and Michael J. Franklin. Clamshell: Speeding up crowds for low-latency data labeling. *Proc. VLDB Endow.*, 9(4):372–383, December 2015. ISSN 2150-8097.

[13] Moritz Hardt and Eric Price. Sharp bounds for learning a mixture of two gaussians. *ArXiv e-prints*, 1404, 2014.

[14] Kevin Jamieson and Robert Nowak. Best-arm identification algorithms for multi-armed bandits in the fixed confidence setting. In *Information Sciences and Systems (CISS)*, pages 1–6. IEEE, 2014.

[15] Matthew L Malloy, Gongguo Tang, and Robert D Nowak. Quickest search for a rare distribution. In *Information Sciences and Systems (CISS)*, pages 1–6. IEEE, 2012.

[16] MTurk. Amazon Mechanical Turk. https://www.mturk.com/.

[17] David Pollard. Asymptopia. *Manuscript in progress. Available at http://www. stat.yale.edu/~pollard*, 2000.

[18] David Siegmund. *Sequential analysis: tests and confidence intervals*. Springer Science & Business Media, 2013.

[19] Robert Spira. Calculation of the gamma function by stirling's formula. *mathematics of computation*, pages 317–322, 1971.

[20] Gautam Thatte, Urbashi Mitra, and John Heidemann. Parametric methods for anomaly detection in aggregate traffic. *IEEE/ACM Trans. Netw.*, 19(2):512–525, April 2011. ISSN 1063-6692.

[21] Yizao Wang, Jean yves Audibert, and Rémi Munos. Algorithms for infinitely many-armed bandits. In D. Koller, D. Schuurmans, Y. Bengio, and L. Bottou, editors, *Advances in Neural Information Processing Systems 21*, pages 1729–1736. Curran Associates, Inc., 2009.

